# Fast Rates to Bayes for Kernel Methods

**Ingo Steinwart**∗ **and Clint Scovel**
Modeling, Algorithms and Informatics Group, CCS-3
Los Alamos National Laboratory
{ingo,jcs}@lanl.gov

## Abstract

We establish learning rates to the Bayes risk for support vector machines (SVMs) with hinge loss. In particular, for SVMs with Gaussian RBF kernels we propose a geometric condition for distributions which can be used to determine approximation properties of these kernels. Finally, we compare our methods with a recent paper of G. Blanchard et al..

## 1  Introduction

In recent years support vector machines (SVM's) have been the subject of many theoretical considerations. In particular, it was recently shown ([1], [2], and [3]) that SVM's can learn for all data-generating distributions. However, these results are purely asymptotic, i.e. no performance guarantees can be given in terms of the number $n$ of samples. In this paper we will establish such guarantees. Since by the no-free-lunch theorem of Devroye (see [4]) performance guarantees are impossible without assumptions on the data-generating distribution we will restrict our considerations to specific classes of distributions. In particular, we will present a geometric condition which describes how distributions behave close to the decision boundary. This condition is then used to establish learning rates for SVM's. To obtain learning rates faster than $n^{-1/2}$ we also employ a noise condition of Tsybakov (see [5]). Combining both concepts we are in particular able to describe distributions such that SVM's with Gaussian kernel learn almost linearly, i.e. with rate $n^{-1+\varepsilon}$ for all $\varepsilon > 0$, even though the Bayes classifier cannot be represented by the SVM.

Let us now formally introduce the statistical classification problem. To this end assume that $X$ is a set. We write $Y := \{-1, 1\}$. Given a *training set* $T = \big((x_1, y_1), \ldots, (x_n, y_n)\big) \in (X \times Y)^n$ the classification task is to predict the *label* $y$ of a new sample $(x, y)$. In the standard batch model it is assumed that $T$ is i.i.d. according to an unknown probability measure $P$ on $X \times Y$. Furthermore, the new sample $(x, y)$ is drawn from $P$ independently of $T$. Given a *classifier* $\mathcal{C}$ that assigns to every training set $T$ a measurable function $f_T : X \to \mathbb{R}$ the prediction of $\mathcal{C}$ for $y$ is $\operatorname{sign} f_T(x)$, where we choose a fixed definition of $\operatorname{sign}(0) \in \{-1, 1\}$. In order to "learn" from the samples of $T$ the decision function $f_T$ should guarantee a small probability for the misclassification of the example $(x, y)$. To make this precise the risk of a measurable function $f : X \to \mathbb{R}$ is defined by

$$\mathcal{R}_P(f) := P\big(\{(x, y) : \operatorname{sign} f(x) \neq y\}\big) \,.$$

The smallest achievable risk $\mathcal{R}_P := \inf\big\{\mathcal{R}_P(f) \mid f : X \to \mathbb{R} \text{ measurable}\big\}$ is called the *Bayes risk* of $P$. A function $f_P : X \to Y$ attaining this risk is called a *Bayes decision function*. Obviously, a good classifier should produce decision functions whose risks are close

to the Bayes risk. This leads to the definition: a classifier is called *universally consistent* if

$$\mathbb{E}_{T \sim P^n} \mathcal{R}_P(f_T) - \mathcal{R}_P \ \to \ 0 \qquad (1)$$

holds for *all* probability measures $P$ on $X \times Y$. The next naturally arising question is whether there are classifiers which guarantee a specific rate of convergence in (1) for *all* distributions. Unfortunately, this is impossible by the so-called no-free-lunch theorem of Devroye (see [4, Thm. 7.2]). However, if one restricts considerations to certain smaller classes of distributions such rates exist for various classifiers, e.g.:

- Assuming that the conditional probability $\eta(x) := P(1|x)$ satisfies certain smoothness assumptions Yang showed in [6] that some plug-in rules (cf. [4]) achieve rates for (1) which are of the form $n^{-\alpha}$ for some $0 < \alpha < 1/2$ depending on the assumed smoothness. He also showed that these rates are optimal in the sense that no classifier can obtain faster rates under the proposed smoothness assumptions.

- It is well known (see [4, Sec. 18.1]) that using structural risk minimization over a sequence of hypothesis classes with finite VC-dimension every distribution which has a Bayes decision function in one of the hypothesis classes can be learned with rate $n^{-1/2}$.

- Let $P$ be a noise-free distribution, i.e. $\mathcal{R}_P = 0$ and $\mathcal{F}$ be a class with finite VC-dimension. If $\mathcal{F}$ contains a Bayes decision function then up to a logarithmic factor the convergence rate of the ERM classifier over $\mathcal{F}$ is $n^{-1}$ (see [4, Sec. 12.7]).

Restricting the class of distributions for classification always raises the question of whether it is likely that these restrictions are met in real world problems. Of course, in general this question cannot be answered. However, experience shows that the assumption that the distribution is noise-free is almost never satisfied. Furthermore, it is rather unrealistic to assume that a Bayes decision function can be represented by the algorithm. Finally, assuming that the conditional probability is smooth, say $k$-times continuously differentiable, seems to be unjustifiable for many real world classification problems. We conclude that the above listed rates are established for situations which are rarely met in practice.

Considering the ERM classifier and hypothesis classes $\mathcal{F}$ containing a Bayes decision function there is a large gap in the rates for noise-free and noisy distributions. In [5] Tsybakov proposed a condition on the noise which describes intermediate situations. In order to present this condition we write $\eta(x) := P(y = 1|x)$, $x \in X$, for the conditional probability and $P_X$ for the marginal distribution of $P$ on $X$. Now, the noise in the labels can be described by the function $|2\eta - 1|$. Indeed, in regions where this function is close to 1 there is only a small amount of noise, whereas function values close to 0 only occur in regions with a high noise. We will use the following modified version of Tsybakov's noise condition which describes the size of the latter regions:

**Definition 1.1** Let $0 \leq q \leq \infty$ and $P$ be a distribution on $X \times Y$. We say that $P$ *has Tsybakov noise exponent* $q$ if there exists a constant $C > 0$ such that for all sufficiently small $t > 0$ we have

$$P_X\big(|2\eta - 1| \leq t\big) \ \leq \ C \cdot t^q . \qquad (2)$$

All distributions have at least noise exponent 0. In the other extreme case $q = \infty$ the conditional probability $\eta$ is bounded away from $\frac{1}{2}$. In particular this means that noise-free distributions have exponent $q = \infty$. Finally note, that Tsybakov's original noise condition assumed $P_X(f \neq f_P) \leq c(\mathcal{R}_P(f) - \mathcal{R}_P)^{\frac{q}{1+q}}$ for all $f : X \to Y$ which is satisfied if e.g. (2) holds (see [5, Prop. 1]).

In [5] Tsybakov showed that if $P$ has a noise exponent $q$ then ERM-type classifiers can obtain rates in (1) which are of the form $n^{-\frac{q+1}{q+pq+2}}$, where $0 < p < 1$ measures the complexity of the hypothesis class. In particular, rates faster than $n^{-1/2}$ are possible whenever $q > 0$ and $p < 1$. Unfortunately, the ERM-classifier he considered is usually hard to implement and in general there exists no efficient algorithm. Furthermore, his classifier requires substantial knowledge on *how* to approximate the Bayes decision rules of the considered distributions. Of course, such knowledge is rarely present in practice.

## 2   Results

In this paper we will use the Tsybakov noise exponent to establish rates for SVM's which are very similar to the above rates of Tsybakov. We begin by recalling the definition of SVM's. To this end let $H$ be a reproducing kernel Hilbert space (RKHS) of a kernel $k : X \times X \to \mathbb{R}$, i.e. $H$ is a Hilbert space consisting of functions from $X$ to $\mathbb{R}$ such that the evaluation functionals are continuous, and $k$ is symmetric and positive definite (see e.g. [7]). Throughout this paper we assume that $X$ is a compact metric space and that $k$ is continuous, i.e. $H$ contains only continuous functions. In order to avoid cumbersome notations we additionally assume $\|k\|_\infty \leq 1$. Now given a regularization parameter $\lambda > 0$ the decision function of an SVM is

$$(f_{T,\lambda}, b_{T,\lambda}) := \arg\min_{\substack{f \in H \\ b \in \mathbb{R}}} \lambda \|f\|_H^2 + \frac{1}{n} \sum_{i=1}^n l\big(y_i(f(x_i) + b)\big),  \tag{3}$$

where $l(t) := \max\{0, 1 - t\}$ is the so-called hinge loss. Unfortunately, only a few results on learning rates for SVM's are known: In [8] it was shown that SVM's can learn with linear rate if the distribution is noise-free and the two classes can be strictly separated by the RKHS. For RKHS which are dense in the space $C(X)$ of continuous functions the latter condition is satisfied if the two classes have strictly positive distance in the input space. Of course, these assumptions are far too strong for almost all real-world problems. Furthermore, Wu and Zhou (see [9]) recently established rates under the assumption that $\eta$ is contained in a Sobolev space. In particular, they proved rates of the form $(\log n)^{-p}$ for some $p > 0$ if the SVM uses a Gaussian kernel. Obviously, these rates are much too slow to be of practical interest and the difficulties with smoothness assumptions have already been discussed above.

For our first result, which is much stronger than the above mentioned results, we need to introduce two concepts both of which deal with the involved RKHS. The first concept describes how well a given RKHS $H$ can approximate a distribution $P$. In order to introduce it we define the *l-risk* of a function $f : X \to \mathbb{R}$ by $\mathcal{R}_{l,P}(f) := \mathbb{E}_{(x,y)\sim P} l(yf(x))$. The smallest possible $l$-risk is denoted by $\mathcal{R}_{l,P} := \inf\{\mathcal{R}_{l,P}(f) \mid f : X \to \mathbb{R}\}$. Furthermore, we define the *approximation error function* by

$$a(\lambda) := \inf_{f \in H} \Big(\lambda \|f\|_H^2 + \mathcal{R}_{l,P}(f)\Big) - \mathcal{R}_{l,P}, \qquad \lambda \geq 0.  \tag{4}$$

The function $a(.)$ quantifies how well an infinite sample SVM with RKHS $H$ approximates the minimal $l$-risk (note that we omit the offset $b$ in the above definition for simplicity). If $H$ is dense in the space of continuous functions $C(X)$ then for *all* $P$ we have $a(\lambda) \to 0$ if $\lambda \to 0$ (see [3]). However, in non-trivial situations no rate of convergence which uniformly holds for all distributions $P$ is possible. The following definition characterizes distributions which guarantee certain polynomial rates:

**Definition 2.1** Let $H$ be a RKHS over $X$ and $P$ be a distribution on $X \times Y$. Then $H$ *approximates $P$ with exponent* $\beta \in (0, 1]$ if there is a $C > 0$ such that for all $\lambda > 0$:

$$a(\lambda) \leq C\lambda^\beta.$$

It can be shown (see [10]) that the extremal case $\beta = 1$ is equivalent to the fact that the minimal $l$-risk can be achieved by an element of $H$. Because of the specific structure of the approximation error function values $\beta > 1$ are only possible for distributions with $\eta \equiv \frac{1}{2}$.

Finally, we need a complexity measure for RKHSs. To this end let $A \subset E$ be a subset of a Banach space $E$. Then the *covering numbers* of $A$ are defined by

$$\mathcal{N}(A, \varepsilon, E) := \min\left\{n \geq 1 : \exists x_1, \ldots, x_n \in E \text{ with } A \subset \bigcup_{i=1}^{n} (x_i + \varepsilon B_E)\right\}, \qquad \varepsilon > 0,$$

where $B_E$ denotes the closed unit ball of $E$. Now our complexity measure is:

**Definition 2.2** Let $H$ be a RKHS over $X$ and $B_H$ its closed unit ball. Then $H$ *has complexity exponent* $0 < p \leq 2$ if there is an $a_p > 0$ such that for all $\varepsilon > 0$ we have

$$\log \mathcal{N}(B_H, \varepsilon, C(X)) \leq a_p \varepsilon^{-p}.$$

Note, that in [10] the complexity exponent was defined in terms of $\mathcal{N}(B_H, \varepsilon, L_2(T_X))$, where $L_2(T_X)$ is the $L_2$-space with respect to the empirical measure of $(x_1, \ldots, x_n)$. Since we always have $\mathcal{N}(B_H, \varepsilon, L_2(T)) \leq \mathcal{N}(B_H, \varepsilon, C(X))$ the Definition 2.2 is stronger than the one in [10]. Here, we only used Definition 2.2 since it enables us to compare our results with [11]. However, all results remain true if one uses the original definition of [10].

For many RKHSs bounds on the complexity exponents are known (see e.g. [3] and [10]). Furthermore, many SVMs use a parameterized family of RKHSs. For such SVMs the constant $a_p$ may play a crucial role. We will see below, that this is in particular true for SVMs using a family of Gaussian RBF kernels. Let us now formulate our first result on rates:

**Theorem 2.3** *Let $H$ be a RKHS of a continuous kernel on $X$ with complexity exponent $0 < p < 2$, and let $P$ be a probability measure on $X \times Y$ with Tsybakov noise exponent $0 < q \leq \infty$. Furthermore, assume that $H$ approximates $P$ with exponent $0 < \beta \leq 1$. We define $\lambda_n := n^{-\frac{4(q+1)}{(2q+pq+4)(1+\beta)}}$. Then for all $\varepsilon > 0$ there is a constant $C > 0$ such that for all $x \geq 1$ and all $n \geq 1$ we have*

$$\mathrm{Pr}^*\left(T \in (X \times Y)^n : \mathcal{R}_P(f_{T,\lambda_n} + b_{T,\lambda_n}) \leq \mathcal{R}_P + Cx^2 n^{-\frac{4\beta(q+1)}{(2q+pq+4)(1+\beta)}+\varepsilon}\right) \geq 1 - e^{-x}.$$

*Here $\mathrm{Pr}^*$ denotes the outer probability of $P^n$ in order to avoid measurability considerations.*

**Remark 2.4** With a tail bound of the form of Theorem 2.3 one can easily get rates for (1). In the case of Theorem 2.3 these rates have the form $n^{-\frac{4\beta(q+1)}{(2q+pq+4)(1+\beta)}+\varepsilon}$ for all $\varepsilon > 0$.

**Remark 2.5** For brevity's sake our major aim was to show the best possible rates using our techniques. Therefore, Theorem 2.3 states rates for the SVM under the assumption that $(\lambda_n)$ is optimally chosen. However, we emphasize, that the techniques of [10] also give rates if $(\lambda_n)$ is chosen in a different (and thus sub-optimal) way. This is also true for our results on SVM's using Gaussian kernels which we will establish below.

**Remark 2.6** In [5] it is assumed that a Bayes classifier is contained in the function class the algorithm minimizes over. This assumption corresponds to a perfect approximation of $P$ by $H$, i.e. $\beta = 1$. In this case our rate is (essentially) of the form $n^{-\frac{2(q+1)}{2q+pq+4}}$. If we rescale the complexity exponent $p$ from $(0, 2)$ to $(0, 1)$ and write $p'$ for the new complexity exponent this rate becomes $n^{-\frac{q+1}{q+p'q+2}}$. This is exactly the *form* of Tsybakov's result in [5]. However, as far as we know our complexity measure cannot be compared to Tsybakov's.

**Remark 2.7** By the nature of Theorem 2.3 it suffices that $P$ satisfies Tsybakov's noise assumption for every $q' < q$. It also suffices to suppose that $H$ approximates $P$ with exponent $\beta'$ for all $\beta' < \beta$, and that $H$ has complexity exponent $p'$ for all $p' > p$. Now, it is shown in [10] that the RKHS $H$ has an approximation exponent $\beta = 1$ if and only if $H$ contains a minimizer of the $l$-risk. In particular, if $H$ has approximation exponent $\beta$ for all $\beta < 1$ but not for $\beta = 1$ then $H$ does not contain such a minimizer but Theorem 2.3 gives the same result as for $\beta = 1$. If in addition the RKHS consists of $C^\infty$ functions we can choose $p$ arbitrarily close to $0$, and hence we can obtain rates up to $n^{-1}$ even though $H$ does not contain a minimizer of the $l$-risk, that means e.g. a Bayes decision function.

In view of Theorem 2.3 and the remarks concerning covering numbers it is often only necessary to estimate the approximation exponent. In particular this seems to be true for the most popular kernel, that is the Gaussian RBF kernel $k_\sigma(x, x') = \exp(-\sigma^2 \|x - x'\|_2^2)$, $x, x' \in X$ on (compact) subsets $X$ of $\mathbb{R}^d$ with width $1/\sigma$. However, to our best knowledge no non-trivial condition on $\eta$ or $f_P = \text{sign} \circ (2\eta - 1)$ which ensures an approximation exponent $\beta > 0$ for *fixed* width has been established and [12] shows that Gaussian kernels poorly approximate smooth functions. Hence plug-in rules based on Gaussian kernels may perform poorly under smoothness assumptions on $\eta$. In particular, many types of SVM's using other loss functions are plug-in rules and therefore, their approximation properties under smoothness assumptions on $\eta$ may be poor if a Gaussian kernel is used. However, our SVM's are not plug-in rules since their decision functions approximate the Bayes decision function (see [13]). Intuitively, we therefore only need a condition that measures the cost of approximating the "bump" of the Bayes decision function at the "decision boundary". We will now establish such a condition for Gaussian RBF kernels with *varying* widths $1/\sigma_n$. To this end let $X_{-1} := \{x \in X : \eta < \frac{1}{2}\}$ and $X_1 := \{x \in X : \eta > \frac{1}{2}\}$. Recall that these two sets are the classes which have to be learned. Since we are only interested in distributions $P$ having a Tsybakov exponent $q > 0$ we always assume that $X = X_{-1} \cup X_1$ holds $P_X$-almost surely. Now we define

$$
\tau_x := \begin{cases} d(x, X_1), & \text{if } x \in X_{-1}, \\ d(x, X_{-1}), & \text{if } x \in X_1, \\ 0, & \text{otherwise}. \end{cases} \tag{5}
$$

Here, $d(x, A)$ denotes the distance of $x$ to a set $A$ with respect to the Euclidian norm. Note that roughly speaking $\tau_x$ measures the distance of $x$ to the "decision boundary". With the help of this function we can define the following geometric condition for distributions:

**Definition 2.8** Let $X \subset \mathbb{R}^d$ be compact and $P$ be a probability measure on $X \times Y$. We say that $P$ has *geometric noise exponent* $\alpha \in (0, \infty]$ if we have

$$
\int_X \tau_x^{-\alpha d} |2\eta(x) - 1| P_X(dx) < \infty. \tag{6}
$$

Furthermore, $P$ has geometric noise exponent $\infty$ if (6) holds for all $\alpha > 0$.

In the above definition we make neither any kind of smoothness assumption nor do we assume a condition on $P_X$ in terms of absolute continuity with respect to the Lebesgue measure. Instead, the integral condition (6) describes the concentration of the measure $|2\eta - 1| dP_X$ near the decision boundary. The less the measure is concentrated in this region the larger the geometric noise exponent can be chosen. In particular, we have $(x \mapsto \tau_x^{-1}) \in L_\infty(|2\eta - 1| dP_X)$ if and only if the two classes $X_{-1}$ and $X_1$ have strictly positive distance! If (6) holds for some $0 < \alpha < \infty$ then the two classes may "touch", i.e. the decision boundary $\partial X_{-1} \cap \partial X_1$ is nonempty. Using this interpretation we easily can construct distributions which have geometric noise exponent $\infty$ and touching classes. In general for these distributions there is no Bayes classifier in the RKHS $H_\sigma$ of $k_\sigma$ for any $\sigma > 0$.

**Example 2.9** We say that $\eta$ is *Hölder about* $\frac{1}{2}$ with exponent $\gamma > 0$ on $X \subset \mathbb{R}^d$ if there is a constant $c_\gamma > 0$ such that for all $x \in X$ we have

$$|2\eta(x) - 1| \leq c_\gamma \tau_x^\gamma . \tag{7}$$

If $\eta$ is Hölder about $1/2$ with exponent $\gamma > 0$, the graph of $2\eta(x) - 1$ lies in a multiple of the envelope defined by $\tau_x^\gamma$ at the top and by $-\tau_x^\gamma$ at the bottom. To be Hölder about $1/2$ it is sufficient that $\eta$ is Hölder continuous, but it is *not* necessary. A function which is Hölder about $1/2$ can be very irregular away from the decision boundary but it cannot jump across the decision boundary discontinuously. In addition a Hölder continuous function's exponent must satisfy $0 < \gamma \leq 1$ where being Hölder about $1/2$ only requires $\gamma > 0$.
For distributions with Tsybakov exponent such that $\eta$ is Hölder about $1/2$ we can bound the geometric noise exponent. Indeed, let $P$ be a distribution which has Tsybakov noise exponent $q \geq 0$ and a conditional probability $\eta$ which is Hölder about $1/2$ with exponent $\gamma > 0$. Then (see [10]) $P$ has geometric noise exponent $\alpha$ for all $\alpha < \gamma \frac{q+1}{d}$.

For distributions having a non-trivial geometric noise exponent we can now bound the approximation error function for Gaussian RBF kernels:

**Theorem 2.10** *Let $X$ be the closed unit ball of the Euclidian space $\mathbb{R}^d$, and $H_\sigma$ be the RKHS of the Gaussian RBF kernel $k_\sigma$ on $X$ with width $1/\sigma > 0$. Furthermore, let $P$ be a distribution with geometric noise exponent $0 < \alpha < \infty$. We write $a_\sigma(.)$ for the approximation error function with respect to $H_\sigma$. Then there is a $C > 0$ such that for all $\lambda > 0$, $\sigma > 0$ we have*

$$a_\sigma(\lambda) \leq C\left(\sigma^d \lambda + \sigma^{-\alpha d}\right). \tag{8}$$

In order to let the right hand side of (8) converge to zero it is necessary to assume both $\lambda \to 0$ and $\sigma \to \infty$. An easy consideration shows that the fastest rate of convergence can be achieved if $\sigma(\lambda) := \lambda^{-\frac{1}{(\alpha+1)d}}$. In this case we have $a_{\sigma(\lambda)}(\lambda) \leq 2C\lambda^{\frac{\alpha}{\alpha+1}}$. Roughly speaking this states that the family of spaces $H_{\sigma(\lambda)}$ approximates $P$ with exponent $\frac{\alpha}{\alpha+1}$. Note, that we can obtain approximation rates up to linear order in $\lambda$ for sufficiently benign distributions. The price for this good approximation property is, however, an increasing complexity of the hypothesis class $H_{\sigma(\lambda)}$ for $\sigma \to \infty$, i.e. $\lambda \to 0$. The following theorem estimates this in terms of the complexity exponent:

**Theorem 2.11** *Let $H_\sigma$ be the RKHS of the Gaussian RBF kernel $k_\sigma$ on $X$. Then for all $0 < p \leq 2$ and $\delta > 0$, there is a $c_{p,d,\delta} > 0$ such that for all $\varepsilon > 0$ and all $\sigma \geq 1$ we have*

$$\sup_{T \in Z^n} \log \mathcal{N}(B_{H_\sigma}, \varepsilon, L_2(T_X)) \leq c_{p,d,\delta}\, \sigma^{(1-\frac{p}{2})(1+\delta)d} \varepsilon^{-p}.$$

Having established both results for the approximation and complexity exponent we can now formulate our main result for SVM's using Gaussian RBF kernels:

**Theorem 2.12** *Let $X$ be the closed unit ball of the Euclidian space $\mathbb{R}^d$, and $P$ be a distribution on $X \times Y$ with Tsybakov noise exponent $0 < q \leq \infty$ and geometric noise exponent $0 < \alpha < \infty$. We define*

$$\lambda_n := \begin{cases} n^{-\frac{\alpha+1}{2\alpha+1}} & \text{if } \alpha \leq \frac{q+2}{2q} \\ n^{-\frac{2(\alpha+1)(q+1)}{2\alpha(q+2)+3q+4}} & \text{otherwise}, \end{cases}$$

*and $\sigma_n := \lambda_n^{-\frac{1}{(\alpha+1)d}}$ in both cases. Then for all $\varepsilon > 0$ there is a $C > 0$ such that for all $x \geq 1$ and all $n \geq 1$ the SVM using $\lambda_n$ and Gaussian RBF kernel with width $1/\sigma_n$ satisfies*

$$\mathrm{Pr}^*\left(T \in (X \times Y)^n : \mathcal{R}_P(f_{T,\lambda_n} + b_{T,\lambda_n}) \leq \mathcal{R}_P + Cx^2 n^{-\frac{\alpha}{2\alpha+1}+\varepsilon}\right) \geq 1 - e^{-x}$$

*if $\alpha \leq \frac{q+2}{2q}$ and*

$$\mathrm{Pr}^*\left(T \in (X \times Y)^n : \mathcal{R}_P(f_{T,\lambda_n} + b_{T,\lambda_n}) \leq \mathcal{R}_P + Cx^2 n^{-\frac{2\alpha(q+1)}{2\alpha(q+2)+3q+4}+\varepsilon}\right) \geq 1 - e^{-x}$$

*otherwise. If $\alpha = \infty$ the latter holds if $\sigma_n = \sigma$ is a constant with $\sigma > 2\sqrt{d}$.*

Most of the remarks made after Theorem 2.3 also apply to the above theorem up to obvious modifications. In particular this is true for Remark 2.4, Remark 2.5, and Remark 2.7.

## 3   Discussion of a modified support vector machine

Let us now discuss a recent result (see [11]) on rates for the following modification of the original SVM:

$$f_{T,\lambda}^* := \arg\min_{f \in H} \lambda\|f\|_H + \frac{1}{n}\sum_{i=1}^n l\big(y_i f(x_i)\big). \tag{9}$$

Note that unlike in (3) the norm of the regularization term is not squared in (9). To describe the result of [11] we need the following modification of the approximation error function:

$$a^*(\lambda) := \inf_{f \in H}\big(\lambda\|f\|_H + \mathcal{R}_{l,P}(f)\big) - \mathcal{R}_{l,P}, \qquad\qquad \lambda \geq 0. \tag{10}$$

Obviously, $a^*(.)$ plays the same role for (9) as $a(.)$ does for (3). Moreover, it is easy to see that for all $\lambda > 0$ with $\|f_{P,\lambda}\| \geq 1$ we have $a^*(\lambda) \leq a(\lambda)$. Now, a slightly simplified version of the result in [11] reads as follows:

**Theorem 3.1** *Let $H$ be a RKHS of a continuous kernel on $X$ with complexity exponent $0 < p < 2$, and let $P$ be a distribution on $X \times Y$ with Tsybakov noise exponent $\infty$. We define $\lambda_n := n^{-\frac{2}{2+p}}$. Then for all $x \geq 1$ there is a $C_x > 0$ such that for all $n \geq 1$ we have*

$$\mathrm{Pr}^*\left(T \in (X \times Y)^n : \mathcal{R}_P(f_{T,\lambda_n}^*) \leq \mathcal{R}_P + C_x\big(a^*(\lambda_n) + n^{-\frac{2}{2+p}}\big)\right) \geq 1 - e^{-x}.$$

Besides universal constants the exact value of $C_x$ is given in [11]. Also note, that the original result of [11] used the eigenvalue distribution of the integral operator $T_k : L_2(P_X) \to L_2(P_X)$ as a complexity measure. If $H$ has complexity exponent $p$ it can be shown that these eigenvalues decay at least as fast as $n^{-2/p}$. Since we only want to compare Theorem 3.1 with our results we do not state the eigenvalue version of Theorem 3.1.

It was also mentioned in [11] that using the techniques therein it is possible to derive rates for the original SVM. In this case $a^*(\lambda_n)$ has to be replaced by $a(\lambda_n)$ and the stochastic term $n^{-\frac{2}{2+p}}$ has to be replaced by "some more involved term" (see [11, p.10]). Since typically $a^*(.)$ decreases faster than $a(.)$ the authors conclude that using a regularization term $\|.\|$ instead of the original $\|.\|^2$ will "necessarily yield an improved convergence rate" (see [11, p.11]). Let us now show that this conclusion is not justified. To this end let us suppose that $H$ approximates $P$ with exponent $0 < \beta \leq 1$, i.e. $a(\lambda) \leq C\lambda^\beta$ for some $C > 0$ and all $\lambda > 0$. It was shown in [10] that this equivalent to

$$\inf_{\|f\|\leq\lambda^{-1/2}} \mathcal{R}_{l,P}(f) - \mathcal{R}_{l,P} \leq c_1 \lambda^{\frac{\beta}{1-\beta}} \tag{11}$$

for some constant $c_1 > 0$ and all $\lambda > 0$. Furthermore, using the techniques in [10] it is straightforward to show that (11) is equivalent to $a^*(\lambda) \leq c_2 \lambda^{\frac{2\beta}{1-\beta}}$. Therefore, if $H$ approximates $P$ with exponent $\beta$ then the rate in Theorem 3.1 becomes $n^{-\frac{4\beta}{(2+p)(1+\beta)}}$ which

is the rate we established in Theorem 2.3 for the *original* SVM. Although the original SVM (3) and the modification (9) learn with the same rate there is a substantial difference in the way the regularization parameter has to be chosen in order to achieve this rate. Indeed, for the original SVM we have to use $\lambda_n = n^{-\frac{4}{(2+p)(1+\beta)}}$ while for (9) we have to choose $\lambda_n = n^{-\frac{2}{2+p}}$. In other words, since $p$ is known for typical RKHS's but $\beta$ is not, we know the asymptotically optimal choice of $\lambda_n$ for (9) while we *do not* know the corresponding optimal choice for the standard SVM. It is naturally to ask whether a similar observation can be made if we have a Tsybakov noise exponent which is smaller than $\infty$. The answer to this question is "yes" and "no". More precisely, using our techniques in [10] one can show that for $0 < q \leq \infty$ the optimal choice of the regularization parameter in (9) is $\lambda_n = n^{-\frac{2(q+1)}{2q+pq+4}}$ leading to the rate $n^{-\frac{4\beta(q+1)}{(2q+pq+4)(1+\beta)}}$. As for $q = \infty$ this rate coincides with the rate we obtained for the standard SVM. Furthermore, the asymptotically optimal choice of $\lambda_n$ is again independent of the approximation exponent $\beta$. However, it depends on the (typically unknown) noise exponent $q$. This leads to the following important questions:

**Question 1:** Is it easier to find an almost optimal choice of $\lambda$ for (9) than for the standard SVM? And if so, what are the computational requirements of solving (9)?

**Question 2:** Can a similar observation be made for the parametric family of Gaussian RBF kernels used in Theorem 2.12 if $P$ has a non-trivial geometric noise exponent $\alpha$?

## References

[1] I. Steinwart. Support vector machines are universally consistent. *J. Complexity*, 18:768–791, 2002.

[2] T. Zhang. Statistical behaviour and consistency of classification methods based on convex risk minimization. *Ann. Statist.*, 32:56–134, 2004.

[3] I. Steinwart. Consistency of support vector machines and other regularized kernel machines. *IEEE Trans. Inform. Theory*, to appear, 2005.

[4] L. Devroye, L. Györfi, and G. Lugosi. *A Probabilistic Theory of Pattern Recognition*. Springer, New York, 1996.

[5] A.B. Tsybakov. Optimal aggregation of classifiers in statistical learning. *Ann. Statist.*, 32:135–166, 2004.

[6] Y. Yang. Minimax nonparametric classification—part I and II. *IEEE Trans. Inform. Theory*, 45:2271–2292, 1999.

[7] N. Cristianini and J. Shawe-Taylor. *An Introduction to Support Vector Machines*. Cambridge University Press, 2000.

[8] I. Steinwart. On the influence of the kernel on the consistency of support vector machines. *J. Mach. Learn. Res.*, 2:67–93, 2001.

[9] Q. Wu and D.-X. Zhou. Analysis of support vector machine classification. Tech. Report, City University of Hong Kong, 2003.

[10] C. Scovel and I. Steinwart. Fast rates for support vector machines. *Ann. Statist.*, submitted, 2003. http://www.c3.lanl.gov/~ingo/publications/ann-03.ps.

[11] G. Blanchard, O. Bousquet, and P. Massart. Statistical performance of support vector machines. *Ann. Statist.*, submitted, 2004.

[12] S. Smale and D.-X. Zhou. Estimating the approximation error in learning theory. *Anal. Appl.*, 1:17–41, 2003.

[13] I. Steinwart. Sparseness of support vector machines. *J. Mach. Learn. Res.*, 4:1071–1105, 2003.
